# Optimistic Linear Programming gives Logarithmic Regret for Irreducible MDPs

**Ambuj Tewari**
Computer Science Division
Univeristy of California, Berkeley
Berkeley, CA 94720, USA
ambuj@cs.berkeley.edu

**Peter L. Bartlett**
Computer Science Division and Department of Statistics
University of California, Berkeley
Berkeley, CA 94720, USA
bartlett@cs.berkeley.edu

## Abstract

We present an algorithm called Optimistic Linear Programming (OLP) for learning to optimize average reward in an irreducible but otherwise unknown Markov decision process (MDP). OLP uses its experience so far to estimate the MDP. It chooses actions by optimistically maximizing estimated future rewards over a set of next-state transition probabilities that are close to the estimates, a computation that corresponds to solving linear programs. We show that the total expected reward obtained by OLP up to time $T$ is within $C(P) \log T$ of the reward obtained by the optimal policy, where $C(P)$ is an explicit, MDP-dependent constant. OLP is closely related to an algorithm proposed by Burnetas and Katehakis with four key differences: OLP is simpler, it does not require knowledge of the supports of transition probabilities, the proof of the regret bound is simpler, but our regret bound is a constant factor larger than the regret of their algorithm. OLP is also similar in flavor to an algorithm recently proposed by Auer and Ortner. But OLP is simpler and its regret bound has a better dependence on the size of the MDP.

## 1   Introduction

Decision making under uncertainty is one of the principal concerns of Artificial Intelligence and Machine Learning. Assuming that the decision maker or *agent* is able to perfectly observe its own state, uncertain systems are often modeled as Markov decision processes (MDPs). Given complete knowledge of the parameters of an MDP, there are standard algorithms to compute *optimal policies*, i.e., rules of behavior such that some performance criterion is maximized. A frequent criticism of these algorithms is that they assume an explicit description of the MDP which is seldom available. The parameters constituting the description are themselves estimated by simulation or experiment and are thus not known with complete reliability. Taking this into account brings us to the well known *exploration vs. exploitation* trade-off. On one hand, we would like to *explore* the system as well as we can to obtain reliable knowledge about the system parameters. On the other hand, if we keep exploring and never *exploit* the knowledge accumulated, we will not behave optimally.

Given a policy $\pi$, how do we measure its ability to handle this trade-off? Suppose the agent gets a numerical reward at each time step and we measure performance by the accumulated reward over time. Then, a meaningful quantity to evaluate the policy $\pi$ is its *regret* over time. To understand what regret means, consider an omniscient agent who knows all parameters of the MDP accurately and behaves optimally. Let $V_T$ be the expected reward obtained by this agent up to time $T$. Let $V_T^\pi$ denote the corresponding quantity for $\pi$. Then the regret $R_T^\pi = V_T - V_T^\pi$ measures how much $\pi$ is hurt due to its incomplete knowledge of the MDP up to time $T$. If we can show that the regret $R_T^\pi$ grows slowly with time $T$, for all MDPs in a sufficiently big class, then we can safely conclude that $\pi$ is making a judicious trade-off between exploration and exploitation. It is rather remarkable that

for this notion of regret, logarithmic bounds have been proved in the literature [1,2]. This means that there are policies $\pi$ with $R_T^\pi = O(\log T)$. Thus the per-step regret $R_T^\pi / T$ goes to zero very quickly.

Burnetas and Katehakis [1] proved that for any policy $\pi$ (satisfying certain reasonable assumptions) $R_T^\pi \geq C_B(P) \log T$ where they identified the constant $C_B(P)$. This constant depends on the transition function $P$ of the MDP[1]. They also gave an algorithm (we call it BKA) that achieves this rate and is therefore optimal in a very strong sense. However, besides assuming that the MDP is irreducible (see Assumption 1 below) they assumed that the support sets of the transition distributions $p_i(a)$ are known for all state-action pairs. In this paper, we not only get rid of this assumption but our optimistic linear programming (OLP) algorithm is also computationally simpler. At each step, OLP considers certain parameters in the vicinity of the estimates. Like BKA, OLP makes *optimistic* choices among these. But now, making these choices only involves solving *linear programs* (LPs) to maximize linear functions over $L_1$ balls. BKA instead required solving non-linear (though convex) programs due to the use of KL-divergence. Another benefit of using the $L_1$ distance is that it greatly simplifies a significant part of the proof. The price we pay for these advantages is that the regret of OLP is $C(P) \log T$ asymptotically, for a constant $C(P) \geq C_B(P)$. We should note here that a number of algorithms in the literature have been inspired by the "optimism in the face of uncertainty" principle [3]–[7].

The algorithm of Auer and Ortner (we refer to it as AOA) is another logarithmic regret algorithm for irreducible[2] MDPs. AOA does not solve an optimization problem at every time step but only when a confidence interval is halved. But then the optimization problem they solve is more complicated because they find a policy to use in the next few time steps by optimizing over a *set of MDPs*. The regret of AOA is $C_A(P) \log T$ where

$$C_A(P) = c \frac{|S|^5 |A| T_w(P) \kappa(P)^2}{\Delta^*(P)^2} \,, \tag{1}$$

for some universal constant $c$. Here $|S|, |A|$ denote the state and action space size, $T_w(P)$ is the worst case hitting time over deterministic policies (see Eqn. (12)) and $\Delta^*(P)$ is the difference between the long term average return of the best policy and that of the next best policy. The constant $\kappa(P)$ is also defined in terms of hitting times. Under Auer and Ortner's assumption of bounded rewards, we can show that the constant for OLP satisfies

$$C(P) \leq \frac{2|S||A|T(P)^2}{\Phi^*(P)} \,. \tag{2}$$

Here $T(P)$ is the hitting time of an optimal policy is therefore necessarily smaller than $T_w(P)$. We get rid of the dependence on $\kappa(P)$ while replacing $T_w(P)$ with $T(P)^2$. Most importantly, we significantly improve the dependence on the state space size. The constant $\Phi^*(P)$ can roughly be thought of as the minimum (over states) difference between the quality of the best and the second best action (see Eqn. (9)). The constants $\Delta^*(P)$ and $\Phi^*(P)$ are similar though not directly comparable. Nevertheless, note that $C(P)$ depends inversely on $\Phi^*(P)$ not $\Phi^*(P)^2$.

## 2   Preliminaries

Consider an MDP $(S, \mathcal{A}, R, P)$ where $S$ is the set of states, $\mathcal{A} = \cup_{i \in S} A(i)$ is the set of actions ($A(i)$ being the actions available in state $i$), $R = \{r(i, a)\}_{i \in S, a \in A(i)}$ are the rewards and $P = \{p_{i,j}(a)\}_{i,j \in S, a \in A(i)}$ are the transition probabilities. For simplicity of analysis, we assume that the rewards are known to us beforehand. We *do not* assume that we know the support sets of the distributions $p_i(a)$.

The *history* $\sigma_t$ up to time $t$ is a sequence $i_0, k_0, \ldots, i_{t-1}, k_{t-1}, i_t$ such that $k_s \in A(i_s)$ for all $s < t$. A *policy* $\pi$ is a sequence $\{\pi_t\}$ of probability distributions on $A$ given $\sigma_t$ such that $\pi_t(A(s_t)|\sigma_t) = 1$ where $s_t$ denotes the random variable representing the state at time $t$. The set of all policies is denoted by $\Pi$. A *deterministic* policy is simply a function $\mu : S \rightarrow \mathcal{A}$ such that $\mu(i) \in A(i)$. Denote the set of deterministic policies by $\Pi_D$. If $\mathcal{D}$ is a subset of $\mathcal{A}$, let $\Pi(\mathcal{D})$ denote the set of

policies that take actions in $\mathcal{D}$. Probability and expectation under a policy $\pi$, transition function $P$ and starting state $i_0$ will be denoted by $\mathbb{P}_{i_0}^{\pi,P}$ and $\mathbb{E}_{i_0}^{\pi,P}$ respectively. Given history $\sigma_t$, let $N_t(i)$, $N_t(i,a)$ and $N_t(i,a,j)$ denote the number of occurrences of the state $i$, the pair $(i,a)$ and the triplet $(i,a,j)$ respectively in $\sigma_t$.

We make the following irreducibility assumption regarding the MDP.

**Assumption 1.** *For all $\mu \in \Pi_D$, the transition matrix $P^\mu = (p_{i,j}(\mu(i)))_{i,j \in S}$ is irreducible (i.e. it is possible to reach any state from any other state).*

Consider the rewards accumulated by the policy $\pi$ before time $T$,

$$V_T^\pi(i_0, P) := \mathbb{E}_{i_0}^{\pi,P}\Big[\sum_{t=0}^{T-1} r(s_t, a_t)\Big]\,,$$

where $a_t$ is the random variable representing the action taken by $\pi$ at time $t$. Let $V_T(i_0, P)$ be the maximum possible sum of expected rewards before time $T$,

$$V_T(i_0, P) := \sup_{\pi \in \Pi} V_T^\pi(i_0, P)\,.$$

The *regret* of a policy $\pi$ at time $T$ is a measure of how well the expected rewards of $\pi$ compare with the above quantity,

$$R_T^\pi(i_0, P) := V_T(i_0, P) - V_T^\pi(i_0, P)\,.$$

Define the long term average reward of a policy $\pi$ as

$$\lambda_\pi(i_0, P) := \liminf_{T \to \infty} \frac{V_T^\pi(i_0, P)}{T}\,.$$

Under assumption 1, the above limit exists and is independent of the starting state $i_0$. Given a restricted set $\mathcal{D} \subseteq \mathcal{A}$ of actions, the *gain* or the best long term average performance is

$$\lambda(P, \mathcal{D}) := \sup_{\pi \in \Pi(\mathcal{D})} \lambda_\pi(i_0, P)\,.$$

As a shorthand, define $\lambda^*(P) := \lambda(P, \mathcal{A})$.

## 2.1 Optimality Equations

A restricted problem $(P, \mathcal{D})$ is obtained from the original MDP by choosing subsets $D(i) \subseteq A(i)$ and setting $\mathcal{D} = \cup_{i \in S} D(i)$. The transition and reward functions of the restricted problems are simply the restrictions of $P$ and $r$ to $\mathcal{D}$. Assumption 1 implies that there is a *bias* vector $h(P, \mathcal{D}) = \{h(i; P, \mathcal{D})\}_{i \in S}$ such that the gain $\lambda(P, \mathcal{D})$ and bias $h(P, \mathcal{D})$ are the unique solutions to the *average reward optimality equations*:

$$\forall i \in S, \ \lambda(P, \mathcal{D}) + h(i; P, D) = \max_{a \in D(i)} [r(i, a) + \langle p_i(a), h(P, \mathcal{D}) \rangle]\,. \tag{3}$$

We will use $h^*(P)$ to denote $h(P, \mathcal{A})$. Also, denote the infinity norm $\|h^*(P)\|_\infty$ by $H^*(P)$. Note that if $h^*(P)$ is a solution to the optimality equations and $\mathbf{e}$ is the vector of ones, then $h^*(P) + c\mathbf{e}$ is also a solution for any scalar $c$. We can therefore assume $\exists i^* \in S, h^*(i^*; P) = 0$ without any loss of generality.

It will be convenient to have a way to denote the quantity inside the 'max' that appears in the optimality equations. Accordingly, define

$$\mathcal{L}(i, a, p, h) := r(i, a) + \langle p, h \rangle\,,$$
$$\mathcal{L}^*(i; P, \mathcal{D}) := \max_{a \in D(i)} \mathcal{L}(i, a, p_i(a), h(P, D))\,.$$

To measure the degree of suboptimality of actions available at a state, define

$$\phi^*(i, a; P) = \mathcal{L}^*(i; P, \mathcal{A}) - \mathcal{L}(i, a, p_i(a), h^*(P))\,.$$

Note that the optimal actions are precisely those for which the above quantity is zero.

$$O(i; P, \mathcal{D}) := \{a \in D(i) \ : \ \phi^*(i, a; P) = 0\}\,,$$
$$\mathcal{O}(P, \mathcal{D}) := \Pi_{i \in S} O(i; P, \mathcal{D})\,.$$

Any policy in $\mathcal{O}(P, \mathcal{D})$ is an optimal policy, i.e.,

$$\forall \mu \in \mathcal{O}(P, \mathcal{D}), \ \lambda_\mu(P) = \lambda(P, \mathcal{D})\,.$$

## 2.2 Critical pairs

From now on, $\Delta^+$ will denote the probability simplex of dimension determined by context. For a suboptimal action $a \notin O(i; P, \mathcal{A})$, the following set contains probability distributions $q$ such that if $p_i(a)$ is changed to $q$, the quality of action $a$ comes within $\epsilon$ of an optimal action. Thus, $q$ makes $a$ look almost *optimal*:

$$\mathrm{MakeOpt}(i, a; P, \epsilon) := \{q \in \Delta^+ \ : \ \mathcal{L}(i, a, q, h^*(P)) \geq \mathcal{L}^*(i; P, \mathcal{A}) - \epsilon\} \ . \tag{4}$$

Those suboptimal state-action pairs for which MakeOpt is never empty, no matter how small $\epsilon$ is, play a crucial role in determining the regret. We call these *critical* state-action pairs,

$$\mathrm{Crit}(P) := \{(i, a) \ : \ a \notin O(i; P, \mathcal{A}) \wedge (\forall \epsilon > 0, \ \mathrm{MakeOpt}(i, a; P, \epsilon) \neq \emptyset)\} \ . \tag{5}$$

Define the function,

$$J_{i,a}(p; P, \epsilon) := \inf\{\|p - q\|_1^2 \ : \ q \in \mathrm{MakeOpt}(i, a; P, \epsilon)\} \ . \tag{6}$$

To make sense of this definition, consider $p = p_i(a)$. The above infimum is then the least distance (in the $L_1$ sense) one has to move away from $p_i(a)$ to make the suboptimal action $a$ look $\epsilon$-optimal. Taking the limit of this as $\epsilon$ decreases gives us a quantity that also plays a crucial role in determining the regret,

$$K(i, a; P) := \lim_{\epsilon \to 0} J_{i,a}(p_i(a); P, \epsilon) \ . \tag{7}$$

Intuitively, if $K(i, a; P)$ is small, it is easy to confuse a suboptimal action with an optimal one and so it should be difficult to achieve small regret. The constant that multiplies $\log T$ in the regret bound of our algorithm OLP (see Algorithm 1 and Theorem 4 below) is the following:

$$C(P) := \sum_{(i,a) \in \mathrm{Crit}(P)} \frac{2\phi^*(i, a; P)}{K(i, a; P)} \ . \tag{8}$$

This definition might look a bit hard to interpret, so we give an upper bound on $C(P)$ just in terms of the infinity norm $H^*(P)$ of the bias and $\Phi^*(P)$. This latter quantity is defined below to be the minimum degree of suboptimality of a critical action.

**Proposition 2.** *Suppose $A(i) = A$ for all $i \in S$. Define*

$$\Phi^*(P) := \min_{(i,a) \in \mathrm{Crit}(P)} \phi^*(i, a; P) \ . \tag{9}$$

*Then, for any $P$,*

$$C(P) \leq \frac{2|S||A|H^*(P)^2}{\Phi^*(P)} \ .$$

See the appendix for a proof.

## 2.3 Hitting times

It turns out that we can bound the infinity norm of the bias in terms of the hitting time of an optimal policy. For any policy $\mu$ define its *hitting time* to be the worst case expected time to reach one state from another:

$$T_\mu(P) := \max_{i \neq j} \mathbb{E}_j^{\mu, P}[\min\{t > 0 \ : \ s_t = i\}] \ . \tag{10}$$

The following constant is the minimum hitting time among optimal policies:

$$T(P) := \min_{\mu \in \mathcal{O}(P, \mathcal{D})} T_\mu(P) \ . \tag{11}$$

The following constant is defined just for comparison with results in [2]. It is the worst case hitting time over all policies:

$$T_w(P) := \max_{\mu \in \Pi_D} T_\mu(P) \ . \tag{12}$$

We can now bound $C(P)$ just in terms of the hitting time $T(P)$ and $\phi^*(P)$.

**Proposition 3.** *Suppose $A(i) = A$ for all $i \in S$ and that $r(i, a) \in [0, 1]$ for all $i \in S, a \in A$. Then for any $P$,*

$$C(P) \leq \frac{2|S||A|T(P)^2}{\Phi^*(P)} \ .$$

See the appendix for a proof.

# 3 The optimistic LP algorithm and its regret bound

---

**Algorithm 1** Optimistic Linear Programming

---

1: **for** $t = 0, 1, 2, \ldots$ **do**
2:      $s_t \leftarrow$ current state
3:
4:      $\triangleright$ Compute solution for "empirical MDP" excluding "undersampled" actions
5:      $\forall i, j \in S, a \in A(i), \; \hat{p}_{i,j}^t(a) \leftarrow \frac{1 + N_t(i,a,j)}{|A(i)| + N_t(i,a)}$
6:      $\forall i \in S, D_t(i) \leftarrow \{ a \in A(i) \; : \; N_t(i,a) \geq \log^2 N_t(i) \}$
7:      $\hat{h}_t, \hat{\lambda}_t \leftarrow$ solution of the optimality equations (3) with $P = \hat{P}^t, \mathcal{D} = \mathcal{D}_t$
8:
9:      $\triangleright$ Compute indices of *all* actions for the current state
10:      $\forall a \in A(s_t), \; U_t(s_t, a) \leftarrow \sup_{q \in \Delta^+} \{ r(s_t, a) + \langle q, \hat{h}_t \rangle \; : \; \| \hat{p}_{s_t}^t(a) - q \|_1 \leq \sqrt{\frac{2 \log t}{N_t(s_t, a)}} \}$
11:
12:      $\triangleright$ Optimal actions (for the current problem) that are about to become "undersampled"
13:      $\Gamma_t^1 \leftarrow \{ a \in O(s_t; \hat{P}^t, \mathcal{D}_t) \; : \; N_t(s_t, a) < \log^2(N_t(s_t) + 1) \}$
14:
15:      $\triangleright$ The index maximizing actions
16:      $\Gamma_t^2 \leftarrow \arg\max_{a \in A(s_t)} U_t(s_t, a)$
17:
18:      **if** $\Gamma_t^1 = O(s_t; \hat{P}^t, \mathcal{D}_t)$ **then**
19:          $a_t \leftarrow$ any action in $\Gamma_t^1$
20:      **else**
21:          $a_t \leftarrow$ any action in $\Gamma_t^2$
22:      **end if**
23: **end for**

---

Algorithm 1 is the Optimistic Linear Programming algorithm. It is inspired by the algorithm of Burnetas and Katehakis [1] but uses $L_1$ distance instead of $KL$-divergence. At each time step $t$, the algorithm computes the empirical estimates for transition probabilities. It then forms a restricted problem ignoring relatively undersampled actions. An action $a \in A(i)$ is considered "undersampled" if $N_t(i,a) < \log^2 N_t(i)$. The solutions $\hat{h}_t, \hat{\lambda}_t$ might be misleading due to estimation errors. To avoid being misled by empirical samples we compute optimistic "indices" $U_t(s_t, a)$ for all legal actions $a \in A(s_t)$ where $s_t$ is the current state. The index for action $a$ is computed by looking at an $L_1$-ball around the empirical estimate $\hat{p}_{s_t}^t(a)$ and choosing a probability distribution $q$ that maximizes $\mathcal{L}(i, a, q, \hat{h}_t)$. Note that if the estimates were perfect, we would take an action maximizing $\mathcal{L}(i, a, \hat{p}_{s_t}^t(a), \hat{h}_t)$. Instead, we take an action that maximizes the index. There is one case where we are forced not to take an index-maximizing action. It is when all the optimal actions of the current problem are about to become undersampled at the next time step. In that case, we take one of these actions (steps 18–22). Note that both steps 7 and 10 can be done by solving LPs. The LP for solving optimality equations can be found in several textbooks (see, for example, [9, p. 391]). The LP in step 10 is even simpler: the $L_1$ ball has only $2|S|$ vertices and so we can maximize over them efficiently.

Like the original Burnetas-Katehakis algorithm, the modified one also satisfies a logarithmic regret bound as stated in the following theorem. Unlike the original algorithm, OLP does not need to know the support sets of the transition distributions.

**Theorem 4.** *Let $\beta$ denote the policy implemented by Algorithm 1. Then we have, for all $i_0 \in S$ and for all $P$ satisfying Assumption 1,*

$$\limsup_{T \to \infty} \frac{R_T^\beta(i_0, P)}{\log T} \leq C(P) \,,$$

*where $C(P)$ is the MDP-dependent constant defined in (8).*

*Proof.* From Proposition 1 in [1], it follows that

$$R_T^\beta(i_0, P) = \sum_{i \in S} \sum_{a \notin O(i; P, \mathcal{A})} \mathbb{E}_{i_0}^{\beta, P}[N_T(i,a)] \phi^*(i, a; P) + O(1) \,. \tag{13}$$

Define the event

$$A_t := \{\|\hat{h}_t - h^*(P)\|_\infty \le \epsilon \wedge \mathcal{O}(\hat{P}^t, \mathcal{D}_t) \subseteq \mathcal{O}(P)\} . \tag{14}$$

Define,

$$N_T^1(i, a; \epsilon) := \sum_{t=0}^{T-1} \mathbf{1}\left[(s_t, a_t) = (i, a) \wedge A_t \wedge U_t(i, a) \ge \mathcal{L}^*(i; P, \mathcal{A}) - 2\epsilon\right] ,$$

$$N_T^2(i, a; \epsilon) := \sum_{t=0}^{T-1} \mathbf{1}\left[(s_t, a_t) = (i, a) \wedge A_t \wedge U_t(i, a) < \mathcal{L}^*(i; P, \mathcal{A}) - 2\epsilon\right] ,$$

$$N_T^3(\epsilon) := \sum_{t=0}^{T-1} \mathbf{1}\left[\bar{A}_t\right] ,$$

where $\bar{A}_t$ denotes the complement of $A_t$. For all $\epsilon > 0$,

$$N_T(i, a) \le N_T^1(i, a; \epsilon) + N_T^2(i, a; \epsilon) + N_T^3(\epsilon) . \tag{15}$$

The result then follows by combining (13) and (15) with the following three propositions and then letting $\epsilon \to 0$ sufficiently slowly. $\square$

**Proposition 5.** *For all $P$ and $i_0 \in S$, we have*

$$\lim_{\epsilon \to 0} \limsup_{T \to \infty} \sum_{i \in S} \sum_{a \notin O(i; P, \mathcal{A})} \frac{\mathbb{E}_{i_0}^{\beta, P}[N_T^1(i, a; \epsilon)]}{\log T} \phi^*(i, a; P) \le C(P) .$$

**Proposition 6.** *For all $P$, $i_0, i \in S$, $a \notin O(i; P, \mathcal{A})$ and $\epsilon$ sufficiently small, we have*

$$\mathbb{E}_{i_0}^{\beta, P}[N_T^2(i, a; \epsilon)] = o(\log T) .$$

**Proposition 7.** *For all $P$ satisfying Assumption 1, $i_0 \in S$ and $\epsilon > 0$, we have*

$$\mathbb{E}_{i_0}^{\beta, P}[N_T^3(\epsilon)] = o(\log T) .$$

# 4 Proofs of auxiliary propositions

We prove Propositions 5 and 6. The proof of Proposition 7 is almost the same as that of Proposition 5 in [1] and therefore omitted (for details, see Chapter 6 in the first author's thesis [8]). The proof of Proposition 6 is considerably simpler (because of the use of L1 distance rather than KL-divergence) than the analogous Proposition 4 in [1].

*Proof of Proposition 5.* There are two cases depending on whether $(i, a) \in \mathrm{Crit}(P)$ or not. If $(i, a) \notin \mathrm{Crit}(P)$, there is an $\epsilon_0 > 0$ such that $\mathrm{MakeOpt}(i, a; P, \epsilon_0) = \emptyset$. On the event $A_t$ (recall the definition given in (14)), we have $|\langle q, \hat{h}_t \rangle - \langle q, h^*(P) \rangle| \le \epsilon$ for any $q \in \Delta^+$. Therefore,

$$
\begin{aligned}
U_t(i, a) &\le \sup_{q \in \Delta^+} \{r(i, a) + \langle q, \hat{h}_t \rangle\} \\
&\le \sup_{q \in \Delta^+} \{r(i, a) + \langle q, h^*(P) \rangle\} + \epsilon \\
&< \mathcal{L}^*(i; P, \mathcal{A}) - \epsilon_0 + \epsilon \qquad\qquad [\because \mathrm{MakeOpt}(i, a; P, \epsilon_0) = \emptyset] \\
&< \mathcal{L}^*(i; P, \mathcal{A}) - 2\epsilon \text{ provided that } 3\epsilon < \epsilon_0
\end{aligned}
$$

Therefore for $\epsilon < \epsilon_0/3$, $N_T^1(i, a; \epsilon) = 0$.

Now suppose $(i, a) \in \mathrm{Crit}(P)$. The event $U_t(i, a) \ge \mathcal{L}^*(i; P, \mathcal{A}) - 2\epsilon$ is equivalent to

$$\exists q \in \Delta^+ \text{ s.t. } \left(\|\hat{p}_i^t(a) - q\|_1^2 \le \frac{2 \log t}{N_t(i, a)}\right) \wedge \left(r(i, a) + \langle q, \hat{h}_t \rangle \ge \mathcal{L}^*(i; P, \mathcal{A}) - 2\epsilon\right) .$$

On the event $A_t$, we have $|\langle q, \hat{h}_t \rangle - \langle q, h^*(P) \rangle| \le \epsilon$ and thus the above implies

$$\exists q \in \Delta^+ \text{ s.t. } \left(\|\hat{p}_i^t(a) - q\|_1^2 \le \frac{2 \log t}{N_t(i, a)}\right) \wedge \left(r(i, a) + \langle q, h^*(P) \rangle \ge \mathcal{L}^*(i; P, \mathcal{A}) - 3\epsilon\right) .$$

Recalling the definition (6) of $J_{i,a}(p; P, \epsilon)$, we see that this implies

$$J_{i,a}(\hat{p}_i^t(a); P, 3\epsilon) \leq \frac{2\log t}{N_t(i, a)} .$$

We therefore have,

$$
\begin{aligned}
N_T^1(i, a; \epsilon) &\leq \sum_{t=0}^{T-1} \mathbf{1}\left[(s_t, a_t) = (i, a) \wedge J_{i,a}(\hat{p}_i^t(a); P, 3\epsilon) \leq \frac{2\log t}{N_t(i, a)}\right] \\
&\leq \sum_{t=0}^{T-1} \mathbf{1}\left[(s_t, a_t) = (i, a) \wedge J_{i,a}(p_i(a); P, 3\epsilon) \leq \frac{2\log t}{N_t(i, a)} + \delta\right] \quad\quad (16) \\
&\quad + \sum_{t=0}^{T-1} \mathbf{1}\left[(s_t, a_t) = (i, a) \wedge J_{i,a}(p_i(a); P, 3\epsilon) > J_{i,a}(\hat{p}_i^t(a); P, 3\epsilon) + \delta\right]
\end{aligned}
$$

where $\delta > 0$ is arbitrary. Each time the pair $(i, a)$ occurs $N_t(i, a)$ increases by 1, so the first count is no more than

$$\frac{2\log T}{J_{i,a}(p_i(a); P, 3\epsilon) - \delta} . \quad\quad (17)$$

To control the expectation of the second sum, note that continuity of $J_{i,a}$ in its first argument implies that there is a function $f$ such that $f(\delta) > 0$ for $\delta > 0$, $f(\delta) \to 0$ as $\delta \to 0$ and $J_{i,a}(p_i(a); P, 3\epsilon) > J_{i,a}(\hat{p}_i^t(a); P, 3\epsilon) + \delta$ implies that $\|p_i(a) - \hat{p}_i^t(a)\|_1 > f(\delta)$. By a Chernoff-type bound, we have, for some constant $C_1$,

$$\mathbb{P}_{i_0}^{\beta, P}[\|p_i(a) - \hat{p}_i^t(a)\|_1 > f(\delta) \mid N_t(i, a) = m] \leq C_1 \exp(-mf(\delta)^2) .$$

and so the expectation of the second sum is no more than

$$\mathbb{E}_{i_0}^{\beta, P}\left[\sum_{t=0}^{T-1} C_1 \exp(-N_t(i, a)f(\delta)^2)\right] \leq \sum_{m=1}^{\infty} C_1 \exp(-mf(\delta)^2) = \frac{C_1}{1 - \exp(-f(\delta)^2)} . \quad\quad (18)$$

Combining the bounds (17) and (18) and plugging them into (16), we get

$$\mathbb{E}_{i_0}^{\beta, P}[N_T^1(i, a; \epsilon)] \leq \frac{2\log T}{J_{i,a}(p_i(a); P, 3\epsilon) - \delta} + \frac{C_1}{1 - \exp(-f(\delta)^2)} .$$

Letting $\delta \to 0$ sufficiently slowly, we get that for all $\epsilon > 0$,

$$\mathbb{E}_{i_0}^{\beta, P}[N_T^1(i, a; \epsilon)] \leq \frac{2\log T}{J_{i,a}(p_i(a); P, 3\epsilon)} + o(\log T) .$$

Therefore,

$$\lim_{\epsilon \to 0} \limsup_{T \to \infty} \frac{\mathbb{E}_{i_0}^{\beta, P}[N_T^1(i, a; \epsilon)]}{\log T} \leq \lim_{\epsilon \to 0} \frac{2}{J_{i,a}(p_i(a); P, 3\epsilon)} = \frac{2}{K(i, a; P)} ,$$

where the last equality follows from the definition (7) of $K(i, a; P)$. The result now follows by summing over $(i, a)$ pairs in $\mathrm{Crit}(P)$. $\qquad\square$

*Proof of Proposition 6.* Define the event

$$A_t'(i, a; \epsilon) := \{(s_t, a_t) = (i, a) \wedge A_t \wedge U_t(i, a) < \mathcal{L}^*(i; P, \mathcal{A}) - 2\epsilon\} ,$$

so that we can write

$$N_T^2(i, a; \epsilon) = \sum_{t=0}^{T-1} \mathbf{1}\left[A_t'(i, a; \epsilon)\right] . \quad\quad (19)$$

Note that on $A_t'(i, a; \epsilon)$, we have $\Gamma_t^1 \subseteq O(i; \hat{P}^t, \mathcal{D}_t) \subseteq O(i; P, \mathcal{A})$. So, $a \notin O(i; P, \mathcal{A})$. But $a$ was taken at time $t$, so it must have been in $\Gamma_t^2$ which means it maximized the index. Therefore, for all optimal actions $a^* \in O(i; P, \mathcal{A})$, we have, on the event $A_t'(i, a; \epsilon)$,

$$U_t(i, a^*) \leq U_t(i, a) < \mathcal{L}^*(i; P, \mathcal{A}) - 2\epsilon .$$

Since $\mathcal{L}^*(i; P, \mathcal{A}) = r(i, a^*) + \langle p_i(a^*), h^*(P) \rangle$, this implies

$$\forall q \in \Delta^+, \; \|q - \hat{p}_i^t(a^*)\|_1 \leq \sqrt{\frac{2 \log t}{N_t(i, a^*)}} \Rightarrow \langle q, \hat{h}_t \rangle < \langle p_i(a^*), h^*(P) \rangle - 2\epsilon \; .$$

Moreover, on the event $A_t$, $|\langle q, \hat{h}_t \rangle - \langle q, h^*(P) \rangle| \leq \epsilon$. We therefore have, for any $a^* \in O(i; P, \mathcal{A})$,

$$A_t'(i, a; \epsilon) \subseteq \left\{ \forall q \in \Delta^+, \; \|q - \hat{p}_i^t(a)\|_1 \leq \sqrt{\frac{2 \log t}{N_t(i, a)}} \Rightarrow \langle q, h^*(P) \rangle < \langle p_i(a), h^*(P) \rangle - \epsilon \right\}$$

$$\subseteq \left\{ \forall q \in \Delta^+, \; \|q - \hat{p}_i^t(a)\|_1 \leq \sqrt{\frac{2 \log t}{N_t(i, a)}} \Rightarrow \|q - p_i(a)\|_1 > \frac{\epsilon}{\|h^*(P)\|_\infty} \right\}$$

$$\subseteq \left\{ \|\hat{p}_i^t(a) - p_i(a)\|_1 > \frac{\epsilon}{h^*(P)} + \sqrt{\frac{2 \log t}{N_t(i, a)}} \right\}$$

$$\subseteq \bigcup_{m=1}^{t} \left\{ N_t(i, a) = m \wedge \|\hat{p}_i^t(a) - p_i(a)\|_1 > \frac{\epsilon}{\|h^*(P)\|_\infty} + \sqrt{\frac{2 \log t}{N_t(i, a)}} \right\}$$

Using a Chernoff-type bound, we have, for some constant $C_1$,

$$\mathbb{P}_{i_0}^{\beta, P}[\|\hat{p}_i^t(a) - p_i(a)\|_1 > \delta \mid N_t(i, a) = m] \leq C_1 \exp(-m\delta^2/2) \; .$$

Using a union bound, we therefore have,

$$\mathbb{P}_{i_0}^{\beta, P}[A_t'(i, a; \epsilon)] \leq \sum_{m=1}^{t} C_1 \exp\left( -\frac{m}{2} \left( \frac{\epsilon}{\|h^*(P)\|_\infty} + \sqrt{\frac{2 \log t}{m}} \right)^2 \right)$$

$$\leq \frac{C_1}{t} \sum_{m=1}^{\infty} \exp\left( -\frac{m\epsilon^2}{2\|h^*(P)\|_\infty^2} - \frac{\epsilon\sqrt{2m \log t}}{\|h^*(P)\|_\infty} \right) = o\left( \frac{1}{t} \right) \; .$$

Combining this with (19) proves the result. $\qquad\square$

## Footnotes

[1]Notation for MDP parameters is defined in Section 2 below.

[2]Auer & Ortner prove claims for unichain MDPs but their usage seems non-standard. The MDPs they call *unichain* are called *irreducible* in standard textbooks (for example, see [9, p. 348])

## References

[1] Burnetas, A.N. & Katehakis, M.N. (1997) Optimal adaptive policies for Markov decision processes. *Mathematics of Operations Research* **22**(1):222–255

[2] Auer, P. & Ortner, R. (2007) Logarithmic online regret bounds for undiscounted reinforcement learning. *Advances in Neural Information Processing Systems 19*. Cambridge, MA: MIT Press.

[3] Lai, T.L. & Robbins, H. (1985) Asymptotically efficient adaptive allocation rules. *Advances in Applied Mathematics* **6**(1):4–22.

[4] Brafman, R.I. & Tennenholtz, M. (2002) R-MAX - a general polynomial time algorithm for near-optimal reinforcement learning. *Journal of Machine Learning Research* **3**:213–231.

[5] Auer, P. (2002) Using confidence bounds for exploitation-exploration trade-offs. *Journal of Machine Learning Research* **3**:397–422.

[6] Auer, P., Cesa-Bianchi, N. & and Fischer, P. (2002) Finite-time analysis of the multiarmed bandit problem. *Machine Learning* **47**(2-3):235-256.

[7] Strehl, A.L. & Littman, M. (2005) A theoretical analysis of model-based interval estimation. In *Proceedings of the Twenty-Second International Conference on Machine Learning*, pp. 857-864. ACM Press.

[8] Tewari, A. (2007) Reinforcement Learning in Large or Unknown MDPs. PhD thesis, Department of Electrical Engineering and Computer Sciences, University of California at Berkeley.

[9] Puterman, M.L. (1994) *Markov Decision Processes: Discrete Stochastic Dynamic Programming*. New York: John Wiley and Sons.

